# Rapid Quality Estimation of Neural Network Input Representations

**Kevin J. Cherkauer**     **Jude W. Shavlik**
Computer Sciences Department, University of Wisconsin-Madison
1210 W. Dayton St., Madison, WI 53706
{cherkauer,shavlik}@cs.wisc.edu

## Abstract

The choice of an input representation for a neural network can have a profound impact on its accuracy in classifying novel instances. However, neural networks are typically computationally expensive to train, making it difficult to test large numbers of alternative representations. This paper introduces fast quality measures for neural network representations, allowing one to quickly and accurately estimate which of a collection of possible representations for a problem is the best. We show that our measures for ranking representations are more accurate than a previously published measure, based on experiments with three difficult, real-world pattern recognition problems.

## 1   Introduction

A key component of successful artificial neural network (ANN) applications is an input representation that suits the problem. However, ANNs are usually costly to train, preventing one from trying many different representations. In this paper, we address this problem by introducing and evaluating three new measures for quickly estimating ANN input representation quality. Two of these, called *ID3leaves* and *Min(leaves)*, consistently outperform Rendell and Ragavan's (1993) *blurring* measure in accurately ranking different input representations for ANN learning on three difficult, real-world datasets.

## 2   Representation Quality

Choosing good input representations for supervised learning systems has been the subject of diverse research in both connectionist (Cherkauer & Shavlik, 1994; Kambhatla & Leen, 1994) and symbolic paradigms (Almuallim & Dietterich, 1994;

Caruana & Freitag, 1994; John et al., 1994; Kira & Rendell, 1992). Two factors of representation quality are well-recognized in this work: the ability to separate examples of different classes (*sufficiency* of the representation) and the number of features present (representational *economy*). We believe there is also a third important component that is often overlooked, namely the ease of learning an accurate concept under a given representation, which we call *transparency*. We define transparency as the density of concepts that are both accurate (generalize well) *and* simple (of low complexity) in the space of possible concepts under a given input representation and learning algorithm. Learning an accurate concept will be more likely if the concept space is rich in accurate concepts that are also simple, because simple concepts require less search to find and less data to validate.

In this paper, we introduce fast transparency measures for ANN input representations. These are orders of magnitude faster than the *wrapper* method (John et al., 1994), which would evaluate ANN representations by training and testing the ANNs themselves. Our measures are based on the strong assumption that, for a fixed input representation, information about the density of accurate, simple concepts under a (fast) decision-tree learning algorithm will transfer to the concept space of an ANN learning algorithm. Our experiments on three real-world datasets demonstrate that our transparency measures are highly predictive of representation quality for ANNs, implying that the transfer assumption holds surprisingly well for some pattern recognition tasks even though ANNs and decision trees are believed to work best on quite different types of problems (Quinlan, 1994).[1] In addition, our Exper. 1 shows that transparency does not depend on representational sufficiency. Exper. 2 verifies this conclusion and also demonstrates that transparency does not depend on representational economy. Finally, Exper. 3 examines the effects of redundant features on the transparency measures, demonstrating that the *ID3leaves* measure is robust in the face of such features.

## 2.1 Model-Based Transparency Measures

We introduce three new "model-based" measures that estimate representational transparency by sampling instances of roughly accurate concept models from a decision-tree space and measuring their complexities. If simple, accurate models are abundant, the average complexity of the sampled models will be low. If they are sparse, we can expect a higher complexity value.

Our first measure, *avg(leaves)*, estimates the expected complexity of accurate concepts as the average number of leaves in $n$ randomly constructed decision trees that correctly classify the training set:

$$avg(leaves) \equiv \tfrac{1}{n} \sum_{t=1}^{n} leaves(t)$$

where *leaves(t)* is the number of leaves in tree $t$. Random trees are built top-down; features are chosen with uniform probability from those which further partition the training examples (ignoring example class). Tree building terminates when each leaf achieves class purity (i.e., the tree correctly classifies all the training examples). High values of *avg(leaves)* indicate high concept complexity (i.e., low transparency).

The second measure, *min(leaves)*, finds the minimum number of leaves over the $n$ randomly constructed trees instead of the average to reflect the fact that learning systems try to make intelligent, not random, model choices:

$$min(leaves) \equiv \min_{t=1,n} \{leaves(t)\}$$

Table 1: Summary of datasets used.

| Dataset | Examples | Classes | Cross Validation Folds |
|---------|----------|---------|------------------------|
| DNA | 20,000 | 6 | 4 |
| NIST | 3,471 | 10 | 10 |
| Magellan | 625 | 2 | 4 |

The third measure, *ID3leaves*, simply counts the number of leaves in the tree grown by Quinlan's (1986) ID3 algorithm:

$$ID3leaves \equiv leaves(ID3\ tree)$$

We always use the full ID3 tree (100% correct on the training set). This measure assumes the complexity of the concept ID3 finds depends on the density of simple, accurate models in its space and thus reflects the true transparency.

All these measures fix tree training-set accuracy at 100%, so simpler trees imply more accurate generalization (Fayyad, 1994) as well as easier learning. This lets us estimate transparency without the multiplicative additional computational expense of cross validating each tree. It also lets us use all the training data for tree building.

## 2.2 "Blurring" as a Transparency Measure

Rendell and Ragavan (1993) address ease of learning explicitly and present a metric for quantifying it called *blurring*. In their framework, the less a representation requires the use of feature interactions to produce accurate concepts, the more transparent it is. *Blurring* heuristically estimates this by measuring the average information content of a representation's individual features. *Blurring* is equivalent to the (negation of the) average information gain (Quinlan, 1986) of a representation's features with respect to a training set, as we show in Cherkauer and Shavlik (1995).

## 3 Evaluating the Transparency Measures

We evaluate the transparency measures on three problems: DNA (predicting gene reading frames; Craven & Shavlik, 1993), NIST (recognizing handwritten digits; "Fl3" distribution), and Magellan (detecting volcanos in radar images of the planet Venus; Burl et al., 1994).[2] The datasets are summarized in Table 1.

To assess the different transparency measures, we follow these steps for each dataset in Exper. 1 and 2:

1. Construct several different input representations for the problem.
2. Train ANNs using each representation and test the resulting generalization accuracy via cross validation (CV). This gives us a (costly) ground-truth ranking of the relative qualities of the different representations.
3. For each transparency measure, compute the transparency score of each representation. This gives us a (cheap) predicted ranking of the representations from each measure.
4. For each transparency measure, compute Spearman's rank correlation coefficient between the ground-truth and predicted rankings. The higher this correlation, the better the transparency measure predicts the true ranking.

_______________

[2]On these problems, we have found that ANNs generalize 1–6 percentage points better than decision trees using identical input representations, motivating our desire to develop fast measures of ANN input representation quality.

Table 2: User CPU seconds on a Sun SPARCstation 10/30 for the largest representation of each dataset. Parenthesized numbers are standard deviations over 10 runs.

| Dataset | Blurring | ID3leaves | Min/Avg(leaves) | Backprop |
|---------|----------|-----------|-----------------|----------|
| DNA | 1.68 (2.38) | 1,245 (3.96) | 13,444 (56.25) | 212,900 |
| NIST | 2.69 (2.31) | 221 (2.75) | 1,558 ( 5.00) | 501,400 |
| Magellan | 0.21 (0.15) | 1 (0.07) | 12 ( 0.13) | 6,300 |

In Exper. 3 we rank only two representations at a time, so instead of computing a rank correlation in step 4, we just count the number of pairs ranked correctly.

We created input representations (step 1) with an algorithm we call RS ("Representation Selector"). RS first constructs a large pool of plausible, domain-specific Boolean features (5,460 features for DNA, 251,679 for NIST, 33,876 for Magellan). For each CV fold, RS sorts the features by information gain on the entire training set. Then it scans the list, selecting each feature that is not strongly pairwise dependent on any feature already selected according to a standard $\chi^2$ independence test using the $X^2$ statistic.

This produces a single reasonable input representation, $R_1$.[3] To obtain the additional representations needed for the ranking experiments, we ran RS several times with successively smaller subsets of the initial feature pool, created by deleting features whose training-set information gains were above different thresholds. For each dataset, we made nine additional representations of varying qualities, labeled $R_2$–$R_{10}$, numbered from least to most "damaged" initial feature pool.

To get the ground-truth ranking (step 2), we trained feed-forward ANNs with backpropagation using each representation and one output unit per class. We tried several different numbers of hidden units in one layer and used the best CV accuracy among these (Fig. 1, left) to rank each input representation for ground truth.

Each transparency measure also predicted a ranking of the representations (step 3). A CPU time comparison is in Table 2. This table and the experiments below report *min(leaves)* and *avg(leaves)* results from sampling 100 random trees, but sampling only 10 trees (giving a factor 10 speedup) yields similar ranking accuracy.

Finally, in Exper. 1 and 2 we evaluate each transparency measure (step 4) using Spearman's rank correlation coefficient, $r_S = 1 - \frac{6 \cdot \sum_{i=1}^{m} d_i^2}{m(m^2-1)}$, between the ground-truth and predicted rankings ($m$ is the number of representations (10); $d_i$ is the ground-truth rank (an integer between 1 and 10) minus the transparency rank). We evaluate the transparency measures in Exper. 3 by counting the number (out of ten) of representation pairs each measure orders the same as ground truth.

## 4    Experiment 1—Transparency vs. Sufficiency

This experiment demonstrates that our transparency measures are good predictors of representation quality and shows that transparency does not depend on representational sufficiency (ability to separate examples). In this experiment we used transparency to rank ten representations for each dataset and compared the rankings to the ANN ground truth using the rank correlation coefficient. RS created the representations by adding features until each representation could completely separate the training data into its classes. Thus, representational sufficiency was

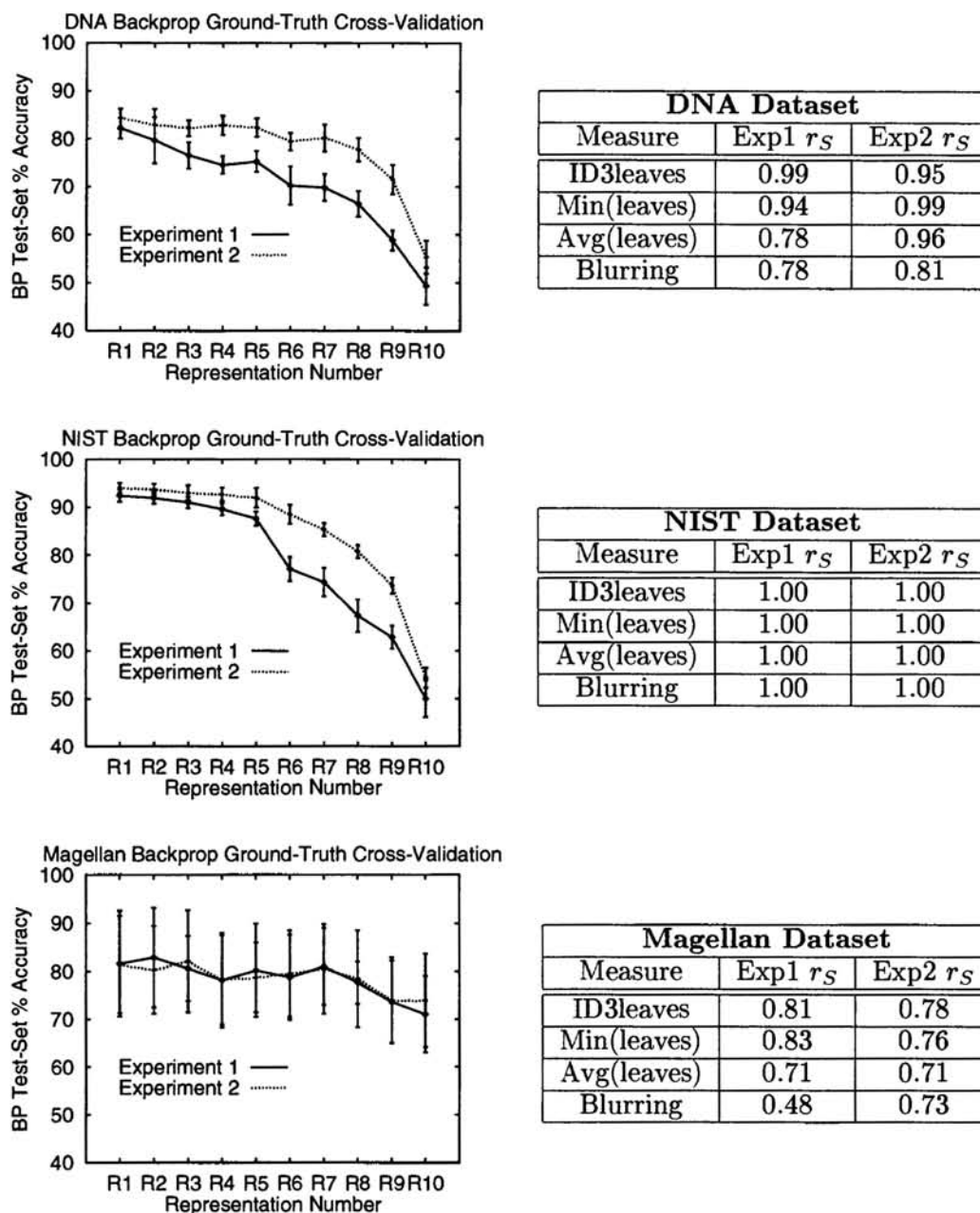

Figure 1: **Left:** Exper. 1 and 2 ANN CV test-set accuracies ($y$ axis; error bars are 1 SD) used to rank the representations ($x$ axis). **Right:** Exper. 1 and 2, transparency rankings compared to ground truth. $r_S$: rank correlation coefficient (see text).

held constant. (The number of features could vary across representations.)

The rank correlation results are shown in Fig. 1 (right). *ID3leaves* and *min(leaves)* outperform the less sophisticated *avg(leaves)* and *blurring* measures on datasets where there is a difference. On the NIST data, all measures produce perfect rankings. The confidence that a true correlation exists is greater than 0.95 for all measures and datasets except *blurring* on the Magellan data, where it is 0.85.

The high rank correlations we observe imply that our transparency measures capture a predictive factor of representation quality. This factor does not depend on representational sufficiency, because sufficiency was equal for all representations.

Table 3: Exper. 3 results: correct rankings (out of 10) by the transparency measures of
the corresponding representation pairs, $R_i$ vs. $R_i'$, from Exper. 1 and Exper. 2.

| Dataset | ID3leaves | Min(leaves) | Avg(leaves) | Blurring |
|---------|-----------|-------------|-------------|----------|
| DNA     | 10        | 0           | 0           | 0        |
| NIST    | 10        | 0           | 0           | 0        |
| Magellan| 6         | 5           | 4           | 4        |

## 5  Experiment 2—Transparency vs. Economy

This experiment shows that transparency does not depend on representational econ-
omy (number of features), and it verifies Exper. 1's conclusion that it does not
depend on sufficiency. It also reaffirms the predictive power of the measures.

In Exper. 1, sufficiency was held constant, but economy could vary. Exper. 2 demon-
strates that transparency does not depend on economy by equalizing the number
of features and redoing the comparison. In Exper. 2, RS added extra features to
each representation used in in Exper. 1 until they all contained a fixed number of
features (200 for DNA, 250 for NIST, 100 for Magellan). Each Exper. 2 represen-
tation, $R_i'$ $(i = 1, ..., 10)$, is thus a proper superset of the corresponding Exper. 1
representation, $R_i$. All representations for a given dataset in Exper. 2 have an
identical number of features and allow perfect classification of the training data, so
neither economy nor sufficiency can affect the transparency scores now.

The results (Fig. 1, right) are similar to Exper. 1's. The notable changes are that
*blurring* is not as far behind *ID3leaves* and *min(leaves)* on the Magellan data as be-
fore, and *avg(leaves)* has joined the accuracy of the other two model-based measures
on the DNA. The confidence that correlations exist is above 0.95 in all cases.

Again, the high rank correlations indicate that transparency is a good predictor
of representation quality. Exper. 2 shows that transparency does not depend on
representational economy or sufficiency, as both were held constant here.

## 6  Experiment 3—Redundant Features

Exper. 3 tests the transparency measures' predictions when the number of redun-
dant features varies, as ANNs can often use redundant features to advantage (Sutton
& Whitehead, 1993), an ability generally not attributed to decision trees.

Exper. 3 reuses the representations $R_i$ and $R_i'$ $(i = 1, ..., 10)$ from Exper. 1 and 2.
Recall that $R_i' \supset R_i$. The extra features in each $R_i'$ are redundant as they are not
needed to separate the training data. We show the number of $R_i$ vs. $R_i'$ representa-
tion pairs each transparency measure ranks correctly for each dataset (Table 3). For
DNA and NIST, the redundant representations always improved ANN generaliza-
tion (Fig. 1, left; 0.05 significance). Only *ID3leaves* predicted this correctly, finding
smaller trees with the increased flexibility afforded by the extra features. The other
measures were always incorrect because the lower quality redundant features de-
graded the random trees *(avg(leaves), min(leaves))* and the average information
gain *(blurring)*. For Magellan, ANN generalization was only significantly different
for one representation pair, and all measures performed near chance.

## 7  Conclusions

We introduced the notion of *transparency* (the prevalence of *simple and accurate*
concepts) as an important factor of input representation quality and developed in-

expensive, effective ways to measure it. Empirical tests on three real-world datasets demonstrated these measures' accuracy at ranking representations for ANN learning at much lower computational cost than training the ANNs themselves. Our next step will be to use transparency measures as scoring functions in algorithms that apply extensive search to find better input representations.

## Acknowledgments

This work was supported by ONR grant N00014-93-1-0998, NSF grant CDA-9024618 (for CM-5 use), and a NASA GSRP fellowship held by KJC.

## Footnotes

[1]We did not preselect datasets based on whether our experiments upheld the transfer assumption. We report the results for all datasets that we have tested our transparency measures on.

[3]Though feature selection is not the focus of this paper, note that similar feature selection algorithms have been used by others for machine learning applications (Baim, 1988; Battiti, 1994).

## References

Almuallim, H. & Dietterich, T. (1994). Learning Boolean concepts in the presence of many irrelevant features. *Artificial Intelligence*, 69(1–2):279–305.

Baim, P. (1988). A method for attribute selection in inductive learning systems. *IEEE Transactions on Pattern Analysis & Machine Intelligence*, 10(6):888–896.

Battiti, R. (1994). Using mutual information for selecting features in supervised neural net learning. *IEEE Transactions on Neural Networks*, 5(4):537–550.

Burl, M., Fayyad, U., Perona, P., Smyth, P., & Burl, M. (1994). Automating the hunt for volcanoes on Venus. In *IEEE Computer Society Conf on Computer Vision & Pattern Recognition: Proc*, Seattle, WA. IEEE Computer Society Press.

Caruana, R. & Freitag, D. (1994). Greedy attribute selection. In *Machine Learning: Proc 11th Intl Conf*, (pp. 28–36), New Brunswick, NJ. Morgan Kaufmann.

Cherkauer, K. & Shavlik, J. (1994). Selecting salient features for machine learning from large candidate pools through parallel decision-tree construction. In Kitano, H. & Hendler, J., eds., *Massively Parallel Artificial Intel*. MIT Press, Cambridge, MA.

Cherkauer, K. & Shavlik, J. (1995). Rapidly estimating the quality of input representations for neural networks. In *Working Notes, IJCAI Workshop on Data Engineering for Inductive Learning*, (pp. 99–108), Montréal, Canada.

Craven, M. & Shavlik, J. (1993). Learning to predict reading frames in *E. coli* DNA sequences. In *Proc 26th Hawaii Intl Conf on System Science*, (pp. 773–782), Wailea, HI. IEEE Computer Society Press.

Fayyad, U. (1994). Branching on attribute values in decision tree generation. In *Proc 12th Natl Conf on Artificial Intel*, (pp. 601–606), Seattle, WA. AAAI/MIT Press.

John, G., Kohavi, R., & Pfleger, K. (1994). Irrelevant features and the subset selection problem. In *Machine Learning: Proc 11th Intl Conf*, (pp. 121–129), New Brunswick, NJ. Morgan Kaufmann.

Kambhatla, N. & Leen, T. (1994). Fast non-linear dimension reduction. In *Advances in Neural Info Processing Sys (vol 6)*, (pp. 152–159), San Francisco, CA. Morgan Kaufmann.

Kira, K. & Rendell, L. (1992). The feature selection problem: Traditional methods and a new algorithm. In *Proc 10th Natl Conf on Artificial Intel*, (pp. 129–134), San Jose, CA. AAAI/MIT Press.

Quinlan, J. (1986). Induction of decision trees. *Machine Learning*, 1:81–106.

Quinlan, J. (1994). Comparing connectionist and symbolic learning methods. In Hanson, S., Drastal, G., & Rivest, R., eds., *Computational Learning Theory & Natural Learning Systems (vol I: Constraints & Prospects)*. MIT Press, Cambridge, MA.

Rendell, L. & Ragavan, H. (1993). Improving the design of induction methods by analyzing algorithm functionality and data-based concept complexity. In *Proc 13th Intl Joint Conf on Artificial Intel*, (pp. 952–958), Chambéry, France. Morgan Kaufmann.

Sutton, R. & Whitehead, S. (1993). Online learning with random representations. In *Machine Learning: Proc 10th Intl Conf*, (pp. 314–321), Amherst, MA. Morgan Kaufmann.